# Approximate Policy Iteration
# with a Policy Language Bias

**Alan Fern and SungWook Yoon and Robert Givan**
Electrical and Computer Engineering, Purdue University, W. Lafayette, IN 47907

## Abstract

We explore approximate policy iteration, replacing the usual cost-function learning step with a learning step in policy space. We give policy-language biases that enable solution of very large *relational* Markov decision processes (MDPs) that no previous technique can solve. In particular, we induce high-quality domain-specific planners for classical planning domains (both deterministic and stochastic variants) by solving such domains as extremely large MDPs.

## 1 Introduction

Dynamic-programming approaches to finding optimal control policies in Markov decision processes (MDPs) [4, 14] using explicit (flat) state space representations break down when the state space becomes extremely large. More recent work extends these algorithms to use propositional [6, 11, 7, 12] as well as relational [8] state-space representations. These extensions have not yet shown the capacity to solve large classical planning problems such as the benchmark problems used in planning competitions [2]. These methods typically calculate a sequence of cost functions. For familiar STRIPS planning domains (among others), useful cost functions can be difficult or impossible to represent compactly.

The above techniques guarantee a certain accuracy at each stage. Here, we focus on inductive techniques that make no such guarantees. Existing inductive forms of approximate policy iteration (API) select compactly represented, approximate cost functions at each iteration of dynamic programming [5], again suffering when such representation is difficult.

We know of no previous work that applies any form of API to benchmark problems from classical planning.[1] Perhaps one reason is the complexity of typical cost functions for these problems, for which it is often more natural to specify a policy space. Recent work on inductive policy selection in relational planning domains [17, 19, 28], has shown that useful policies can be learned using a policy-space bias, described by a generic knowledge representation language. Here, we incorporate that work into a practical approach to API for STRIPS planning domains.

We replace the use of cost-function approximations as policy representations in API[2] with direct, compact state-action mappings, and use a standard relational learner to learn these mappings. We inherit from familiar API methods a (sampled) policy-evaluation phase using simulation of the current policy, or rollout [25], and an inductive policy-selection phase inducing an approximate next policy from sampled current policy values.

We evaluate our API approach in several STRIPS planning domains, showing iterative policy improvement. Our technique solves entire *planning domains*, finding a policy that can be applied to any problem in the domain, rather than solving just a single problem instance from the domain. We view each planning domain as a single large MDP where each "state" specifies both the current world and the goal. The API method thus learns control knowledge (a "policy") for the given planning domain.

Our API technique naturally leverages heuristic functions (cost function estimates), if available—this allows us to benefit from recent advances in domain-independent heuristics for classical planning, as discussed below. Even when greedy heuristic search solves essentially none of the domain instances, our API technique successfully bootstraps from the heuristic guidance. We also demonstrate that our technique is able to iteratively improve policies that correspond to previously published hand-coded control knowledge (for TL-plan [3]) and policies learned by Yoon et al. [28]. Our technique gives a new way of using heuristics in planning domains, complementing traditional heuristic search strategies.

## 2  Approximate Policy Iteration

We first review API for a general, action-simulator–based MDP representation, and later, in Section 3, detail a particular representation of planning domains as relational MDPs and the corresponding policy-space learning bias.

**Problem Setup.** We follow and adapt [16] and [5]. We represent an MDP using a generative model $\langle S, A, T, C, I \rangle$, where $S$ is a finite set of states, $A$ is a finite set of actions, and $T$ is a randomized "action-simulation" algorithm that, given state $s$ and action $a$, returns a next state $t$. The component $C$ is an action-cost function that maps $S \times A$ to real-numbers, and $I$ is a randomized "initial-state algorithm" with no inputs that returns a state in $S$. We sometimes treat $I$ and $T(s, a)$ as random variables.

For MDP $\mathcal{M} = \langle S, A, T, C, I \rangle$, a policy $\pi$ is a (possibly stochastic) mapping from $S$ to $A$. The *cost function* $J_M^\pi(s)$ and the *Q-cost function* $Q_M^\pi(s, a)$ are the unique solutions to

$$Q_M^\pi(s, a) = C(s, a) + \alpha E[J_M^\pi(T(s, a))], \text{ where } J_M^\pi(s) = E[Q_M^\pi(s, \pi(s))],$$

representing the expected, cumulative, discounted cost of following policy $\pi$ in $M$ starting from state $s$, and where $0 \leq \alpha < 1$ is the discount factor. In this work, we seek to heuristically minimize $E[J_M^\pi(I)]$, due to the complexity of the problems we consider.

Given a current policy $\pi$, we can define a new improved policy $\mathcal{PI}[\pi](s)$ by $\text{argmin}_{a \in A} Q_M^\pi(s, a)$. The cost function of $\mathcal{PI}[\pi]$ is guaranteed to be no worse than that of $\pi$ at each state and to improve at some state for non-optimal $\pi$. *Exact policy iteration* iterates policy improvement ($\mathcal{PI}$) from any initial policy to reach an optimal fixed point. Policy improvement is divided into two steps: computing $J_M^\pi$ (policy evaluation) and then computing $Q_M^\pi$ and selecting the minimizing action (policy selection).

**Approximate Policy Iteration.** API, as described in [5], heuristically approximates policy iteration in large state spaces by using an approximate policy-improvement operator trained with Monte-Carlo simulation. The approximate operator performs policy evaluation by simulation—evaluating a policy $\pi$ at a state $s$ by drawing some number of sample trajectories of $\pi$ starting at $s$—and performs policy selection by constructing a training set of samples of either the $J$ or $Q$ cost functions from a "small" but "representative" set of states and then using this training set to induce a new "approximately improved" policy.

The use of API assumes that states and perhaps actions are represented in factored form (typically, a feature vector) that facilitates generalizing properties of the training data to the entire state and action spaces. Due to API's inductive nature, there are typically no guarantees for policy improvement—nevertheless, API often "converges" usefully, e.g. [24, 26].

We start API by providing it with an initial policy $\pi_0$ and a real-valued heuristic function

$H$, where $H(s)$ is interpreted as an estimate of the cost of state $s$ (presumably with respect to the optimal policy). We note that $H$ or $\pi_0$ may be trivial, i.e. always returning a constant or random action respectively. For API to be effective, however, it is important that $\pi_0$ and $H$ combine to provide guidance toward improvement. For example, in goal-based planning domains either $\pi_0$ should occasionally reach a goal or $H$ should provide non-trivial goal-distance information. In our experiments we consider scenarios that use different types of initial policies and heuristics to bootstrap API.

Given $\pi_0$, $H$, and an MDP $M = \langle S, \{a_1, \ldots, a_m\}, T, C, I \rangle$, API produces a policy sequence by iterating steps of approximate policy improvement—note that $\pi_0$ is used in only the initial iteration but the heuristic is always used. Approximate policy improvement computes an (approximate) improvement $\pi'$ of a policy $\pi$ by attempting to approximate the output of exact policy improvement, i.e. $\pi'(s) = \operatorname{argmin}_{a \in A} Q_M^\pi(s, a)$. There are two steps: estimating $Q$-costs for all actions at a representative set of states, and using resulting data set to learn an approximation of $\pi'$. Figure 1 gives pseudo-code for our variant of API.

**Step 1: $Q$-Cost Estimation via Rollout.** (see [25]) Given $\pi$, we construct a training set $D$, describing an improved policy $\pi'$, consisting of tuples $\langle s, \pi(s), \hat{Q}(s, a_1), \ldots, \hat{Q}(s, a_m) \rangle$. For each sampled state $s$ and action $a$, the term $\hat{Q}(s, a)$ refers to $Q_M^\pi(s, a)$ as estimated by drawing "sampling width" trajectories of length "horizon" from $s$ and computing the average discounted trajectory cost over the sampled trajectories, where the cost of a trajectory includes the value of the heuristic function at the horizon state. To get a "representative set" of states, we include each state $s$ visited by $\pi'$ (as indicated by the $\hat{Q}$ estimates) within "horizon" steps from one of "training set size" states drawn from the initial distribution.[3]

**Step 2: Learn Policy.** Select $\pi'$ with the goal of minimizing the cumulative $\hat{Q}$-cost for $\pi'$ over $D$ (approximating the same minimization over $S$ in exact policy iteration). Traditional API uses a cost-function space learning bias in this selection—in Section 3 we detail the policy-space learning bias used by our technique. By labeling each training state with the associated $Q$-costs for each action, rather than simply with the best action, we enable the learner to make more informed trade-offs. We note that the inclusion of $\pi(s)$ in each training example enables the learner to normalize the data, if desired—e.g. our learner (see Section 3) uses a bias that focuses on states where large improvement appears possible.

## 3   API for Relational Planning

In order to use our API framework, we represent classical planning domains (not just single instances) as relationally factored MDPs. We then describe our compact relational policy language and the associated learner for use in step 2 of our API framework.

**Planning Domains as MDPs.** We say that an MDP $\langle S, A, T, C, I \rangle$ is *relational* when $S$ and $A$ are defined by giving the finite sets of objects $O$, predicates $P$, and action types $Y$. A *fact* is a predicate applied to the appropriate number of objects. A state in $S$ is a set of facts (taken to be "true" in the state), and $S$ is all such states. An *action* is an action type applied to the appropriate number of objects, and the action space $A$ is the set of all actions.

A classical planning domain is specified by providing a set of world predicates, action types, and an action simulator. We simultaneously solve all problem instances of such a planning domain[4] by constructing a relational MDP as described below.

Let $O$ be a fixed set of objects and $Y$ be the set of action types from the planning domain. Together, $O$ and $Y$ define the MDP action space. Each MDP state is a single problem

| **API** $(n, w, h, H, \pi_0)$ | **Draw-Training-Set**$(n, w, h, H, \pi)$ |
|---|---|
| // *training set size $n$, sampling width $w$,* <br> // *horizon $h$, initial policy $\pi_0$,* <br> // *cost estimator (heuristic function) $H$.* <br><br> $\pi \leftarrow \pi_0$; <br> **loop** <br>     $D \leftarrow$ **Draw-Training-Set**$(n, w, h, H, \pi)$; <br>     $\pi \leftarrow$ **Learn-Decision-List**$(D)$; <br> **until** satisfied with $\pi$; <br>                *//e.g. until change is small* <br> **Return** $\pi$; | // *training set size $n$, sampling width $w$,* <br> // *horizon $h$, cost estimator $H$, current policy $\pi$* <br><br> $D \leftarrow \emptyset$;   $E \leftarrow$ set of $n$ states sampled from $I$; <br> **for** each state $s_0 \in E$   // *Draw trajectory of* <br>                         // *sample states from $s_0$* <br>    $s \leftarrow s_0$; <br>    **for** $i = 1$ to $h$ <br>      $Q_\pi(s) \leftarrow$ **Policy-Rollout**$(\pi, s, w, h, H)$; <br>      $a \leftarrow$ action maximizing $Q_\pi(s, a)$; <br>      $D \leftarrow \langle s, \pi(s), Q_\pi(s) \rangle \cup D$; <br>      $s \leftarrow$ state sampled from $T(s, a)$; <br> **Return** $D$; |

**Policy-Rollout** $(\pi, s, w, h, H)$    // *Computes estimate of $Q_\pi(s)$*

// *policy $\pi$, state $s$, sampling width $w$, horizon $h$, cost estimator $H$*

Initialize $Q_\pi(s)$, a vector indexed by the actions in $A$, to zeroes;
**for**$_1$ each action $a$ in $A$
     **for**$_2$ *sample* = 1 to $w$
         $s' \leftarrow s$;
         **for**$_3$ *step* = 1 to $h$
             $Q_\pi(s, a) \leftarrow Q_\pi(s, a) + C(s', \pi(s'))$;
             $s' \leftarrow$ a state sampled from $T(s', \pi(s'))$    // *end for$_3$*
         $Q_\pi(s, a) \leftarrow Q_\pi(s, a) + H(s')$;           // *end for$_2$*
     $Q_\pi(s, a) \leftarrow \frac{Q_\pi(s, a)}{w}$                // *end for$_1$*
**Return** $Q_\pi(s)$

Figure 1: Pseudo-code for our API algorithm. The MDP $\langle S, A, T, C, I \rangle$ is assumed globally known. The general approach is inherited from [5], and is restated here for clarity. Key differences are the use of **Learn-Decision-List** [28], as discussed in Section 3, and the choice of action $a$ in **Draw-Training-Set** (see Footnote 3).

instance (i.e. an initial state and a goal) from the planning domain by specifying both the current world and the goal. We achieve this by letting $P$ be the set of world predicates from the classical domain together with a new set of *goal predicates*, one for each world predicate. Goal predicates are named by prepending a 'g' to the corresponding world predicate. Thus, the MDP states are sets of world and goal facts involving some or all objects in $O$.

The objective is to reach MDP states where the goal facts are a subset of the world facts (*goal states*). The state $\{\mathbf{on\text{-}table}(a), \mathbf{on}(a, b), \mathbf{clear}(b), \mathbf{gclear}(b)\}$ is thus a goal state in a blocks-world MDP, but would not be a goal state without $\mathbf{clear}(b)$. We represent this objective by defining $C$ to assign zero cost to actions taken in goal states and a positive cost to actions in all other states. In addition, we take $T$ to be the action simulator from the planning domain (e.g. as defined by STRIPS rules), modified to treat goal states as terminal and to preserve without change all goal predicates. With this cost function, a low-cost policy must arrive at goal states as "quickly" as possible. Finally, the initial state distribution $I$ can be any program that generates legal problem instances (MDP states) of the planning domain—e.g. one might use a problem generator from a planning competition. While here we assume and accurate $T$ model is known, a more general reinforcement-learning context would require learning an approximate $T$, trading off exploitation of this model with exploration to improve it.

**Taxonomic Decision List Policies.** We adapt the API method of Section 2 by using, for Step 2, the policy-space language bias and learning method of our previous work on learning policies in relational domains from small problem solutions [28], briefly reviewed here.

In relational domains, useful rules often take the form "apply action type $a$ to any object in set $C$", e.g. "unload any object that is at its destination". In [19], decision lists of such rules were used as a language bias for learning policies. We use such lists, and represent the sets of objects needed using *class expressions* $C$ written in taxonomic syntax [20], defined by

$$C ::= C_0 \mid \textbf{anything} \mid \neg C \mid (R\ C) \mid C \cap C, \text{ with } R ::= R_0 \mid R^{-1} \mid R \cap R \mid R^*.$$

Here, $C_0$ is any one argument relation and $R_0$ any binary relation from the predicates in $P$. One argument relations denote the set of objects that they are true of, $(R\ C)$ denotes the image of the objects in class $C$ under the binary relation $R$, and for the (natural) semantics of the other constructs shown, please refer to [28]. Given a state $s$ and a concept $C$ expressed in taxonomic syntax, it is straightforward to compute, in time polynomial in the sizes of $s$ and $C$, the set of domain objects that are represented by $C$ in $s$.

Restricting our attention to one-argument–action types[5], we write a policy as $\langle C_1{:}a_1, C_2{:}a_2, \ldots, C_n{:}a_n \rangle$, where the $C_i$ are taxonomic-syntax concepts and the $a_i$ are action types. See Yoon et al. [28] for examples and details.

Our learner builds a decision-list of size-bounded rules by starting with the empty list and greedily selecting a new rule to add, continuing until the list "covers" all of the training data. This procedure is described in Yoon et al. [28], where a heuristically guided beam-search is used to greedily select the next rule to add. The only difference between the learner in [28] and the one used here is the heuristic function, which incorporates Q-cost information (unlike [28]). Given training example $\langle s, \pi\,(s), \hat{Q}(s, a_1), \ldots, \hat{Q}(s, a_m) \rangle$ in $D$, we define the Q-advantage of taking action $a$ instead of $\pi(s)$ in state $s$ by $\Delta(s, a) = \hat{Q}(s, \pi\,(s)) - \hat{Q}(s, a)$. We take the heuristic value of a concept-action rule to be the number of training examples where the rule "fires" plus the cumulative Q-advantage that the rule achieves on those training examples.[6] Using Q-advantage rather than Q-cost focuses the learner toward instances where large improvement over the previous policy is possible.

## 4   Relational Planning Experiments

Our experiments support three claims. 1) Using only the guidance of an (often weak) domain-independent heuristic, API learns effective policies for entire classical planning domains. 2) Each learned policy is a domain-specific planner that is fast and empirically compares well to the state-of-the-art domain-independent planner FF [13]. 3) API can improve on previously published control knowledge and on that learned by previous systems.

**Domains.** We consider two deterministic domains with standard definitions and three stochastic domains from Yoon et al. [28]—these are: BW($n$), the $n$-block blocks world; LW($l$,$t$,$p$), the $l$ location, $t$ truck, $p$ package logistics world; SBW($n$), a stochastic variant of BW($n$); SLW($l$,$c$,$t$,$p$), the stochastic logistics world with $c$ cars and $t$ trucks; and SPW($n$), a version of SBW($n$) with a paint action. We draw problem instances from each domain by generating pairs of random initial states and goal conditions. The goal conditions specify block configurations involving all blocks in blocks worlds, and destinations for all packages in logistics worlds.[7]

Throughout, we use the domain-independent FF heuristic [13].[8] Each experiment specifies a planning domain and an initial policy and then iterates API[9] until "no more progress" is made. We evaluate each policy on 1000 random problem instances, recording the *success*

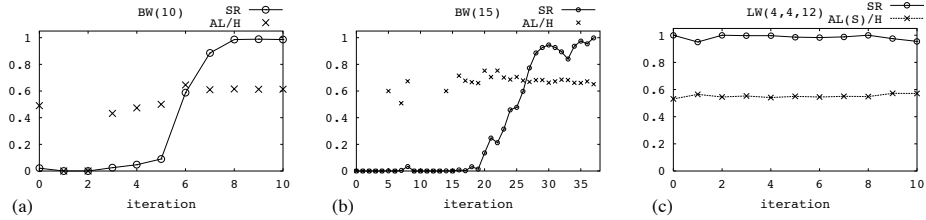

Figure 2: Bootstrapping API with a domain-independent heuristic.

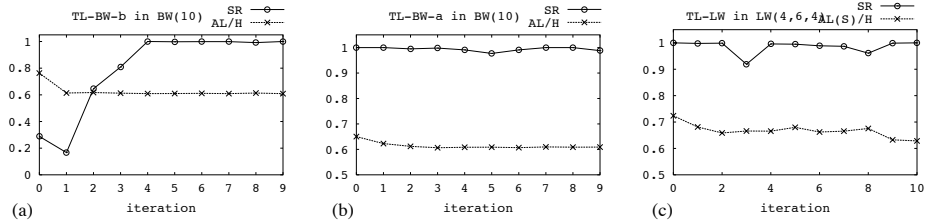

Figure 3: Using TL-Plan control knowledge as initial policies.

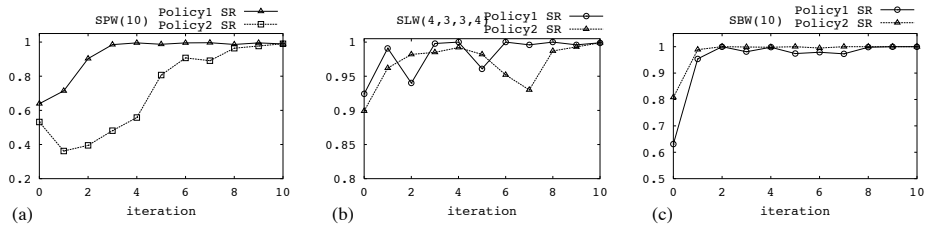

Figure 4: Using previously learned initial policies.

*ratio* SR (fraction of problems solved within the horizon) and *normalized average solution length* AL/H (average plan length in *successful trials* divided by horizon), omitting AL/H for very low SR. Initial-policy performance is plotted at iteration zero.

**Bootstrapping from the Heuristic.** We consider the domain-independent initial policy[10] FF-Greedy, which acts using the FF heuristic with one-step look-ahead. Figures 2a and b show SR and AL/H after each API iteration for BW(10) and BW(15). FF-Greedy is poor in both domains. There is an initial period of no (apparent) progress, followed by rapid improvement to nearly perfect SR. Examination of the learned BW(15) policies shows that early iterations find important concepts and later iterations find a policy that achieves a small SR; at that point, rapid improvement ensues. Figure 2c shows the SR and AL/H for LW(4,4,12). FF-Greedy performs very well here; nevertheless, API yields compact declarative policies of the same quality as FF-Greedy. We replicated these experiments in the stochastic variants of these domains, with similar results (not shown for space reasons).

**Initial Hand-Coded Policies.** TL-Plan [3] uses human-coded domain-specific control knowledge to solve classical planning problems. Here we use initial policies for API that correspond to the domain-specific control knowledge appearing in [3].[11] For the blocks

world TL-Plan provides three sets of control knowledge of increasing quality—we use the best and second best sets to get the policies TL-BW-a and TL-BW-b, respectively. For logistics there is only one set of knowledge given, yielding the policy TL-LW.

Figures 3a–3c show the SR and AL/H for API when starting with TL-BW-a and TL-BW-b in BW(10) and TL-LW in LW(4,4,12). In each case, API improves the human-coded policies. Starting with TL-BW-a and TL-LW, which have perfect SR, API uncovers policies that maintain SR but improve AL/H by approximately 6.3% and 13%, respectively. Starting with TL-BW-b, which has SR of only 30%, API quickly uncovers policies with perfect SR.

There is a dramatic difference in the quality of FF-Greedy (iteration 0 of Figure 2a), TL-BW-a, and TL-BW-b in BW(10); yet, for each initial policy, API finds policies of roughly identical quality—requiring more iterations for lower quality initial policies.

**Initial Machine-Learned Policies.** In Yoon et al. [28], policies were learned from solutions to randomly drawn small problems for the three stochastic domains we test here, among others. A significant range of policy qualities results, due to the random draw. Here, we use API starting with some below-average policies from that work.[12] Figures 4a-c show results for SPW(10), SLW(4,3,3,4), and SBW(10). For each domain, API is shown to improve the SR for two arbitrarily selected, below-average, learned starting policies to nearly 1.0. API successfully exploits the previous, noisy learning to robustly obtain a good policy.

**Comparing learned policies to FF.** A learned policy corresponds to a domain-specific planner for the target planning domain. Here we show that these policies are competitive with FF, a state-of-the-art AI planer, with respect to planning time and success ratio. We selected a blocks-world policy and logistics-world policy corresponding to the learned policies (beyond iteration 0) in Figures 2a and c with the best SR, breaking

Table 1: FF vs. learned policies.

| Domains | FF (in C) | | | API (Scheme) | | |
|---|---|---|---|---|---|---|
| | SR | AL | Time | SR | AL | Time |
| BW(10) | 1 | 33 | 0.1s | 0.99 | 25 | 1.5s |
| BW(15) | 0.96 | 58 | 2.7s | 0.99 | 39 | 2.5s |
| BW(20) | 0.75 | 62 | 27.7s | 0.98 | 55 | 3.7s |
| BW(30) | 0.14 | 103 | 166.0s | 0.99 | 86 | 2.8s |
| LW(4,4,12) | 1 | 42 | 0.0s | 1 | 43 | 2.7s |
| LW(5,14,20) | 1 | 73 | 0.4s | 1 | 74 | 3.6s |

ties with AL. We applied FF and the appropriate selected policy to each of 1000 new test problems from each of the domains shown in Table 1. Planning cutoff times were set at 600, 300, and 100 seconds for BW(30), BW(20), and all other domains, respectively. Table 1 records the percent of problems solved within the time cutoff (SR), the average length of *successful* trials (AL), and the average time for *successful* trials (Time) for both FF and our two selected policies.

In blocks worlds with more than 10 blocks, the API policy improves on FF in every category, with scaling much better to 20 and 30 blocks. Using the same heuristic information (in a different way), API uncovers policies that significantly outperform FF. FF's heuristic is well suited to logistics worlds, eliminating search for these problems. Our method performs equivalently, but for the slow prototype Scheme implementation.

## 5   Related Work

Typically, previous "learning for planning" systems [22] learn from small-problem solutions to improve the efficiency and/or quality of planning. Two primary approaches are to learn control knowledge for search-based planners, e.g. [23, 27, 10, 15, 1], and, more closely related, to learn stand-alone control policies [17, 19, 28].

The former work is severely limited by the utility problem (see [21]), i.e., being "swamped" by low utility rules. Critically, our policy-language bias confronts this issue by preferring simpler policies. Regarding the latter, our work is novel in using API to iteratively improve

policies, and leads to a more robust learner, as shown above. In addition, we leverage a domain-independent planning heuristic to avoid the need for access to small problems. Our learning approach is also not tied to having a base planner.

The most closely related work is relational reinforcement learning (RRL) [9], a form of online API that learns relational cost-function approximations. $Q$-cost functions are learned in the form of relational decision trees ($Q$-trees) and are used to learn corresponding policies ($P$-trees). The RRL results clearly demonstrate the difficulty of learning cost-function approximations in relational domains. Compared to $P$-trees, $Q$-trees tend to generalize poorly and be much larger. RRL has not yet demonstrated scalability to problems as complex as those considered here—previous RRL blocks-world experiments include relatively simple goals[13], which lead to cost functions that are much less complex than the ones here. However, unlike RRL, our API assumes an unconstrained simulator and (for the FF heuristic) a world model, which must be provided or learned by additional techniques.

## Footnotes

[1]Recent work in *relational reinforcement learning* has been applied to STRIPS problems with much simpler goals than typical benchmark planning domains, and is discussed below in Section 5.

[2]In concurrent work, [18] pursued a similar approach to API in attribute-value domains.

[3]It is important that states are sampled from $\pi'$ rather than $\pi$ to match the training distribution to the implied "test set" distribution.

[4]As an example, the blocks world is a classical planning domain, where a problem instance is an initial block configuration and a set of goal conditions. Classical planners attempt to find solutions to specific problem instances of a domain.

[5]Multiple argument actions can be simulated at some cost with multiple single argument actions.

[6]If the coverage term is not included, then covering a zero Q-advantage example is the same as not covering it. But zero Q-advantage can be good (e.g. the previous policy is optimal in that state).

[7]PSTRIPS domain definitions are at http://www.ece.purdue.edu/~givan/nips03-domains.html.

[8]Space precludes a description of this complex and well studied planning heuristic here.

[9]We use discount factor 1 and select large enough horizons to accurately rank most policies: $4 \times n$ for BW($n$) and SBW($n$), $6 \times n$ for SPW($n$), $12 \times p$ for LW($l$,$t$,$p$) and SLW($l$,$c$,$t$,$p$). Training set size is

[10]100 trajectories, and sampling width is always 1, which worked well even for stochastic domains. A sampling width of 1 corresponds to a preference to draw a small number of trajectories from each of a variety of problems rather than a larger number from each of relatively fewer training problems—in either case, the learner must be robust to the noise resulting from stochastic effects.

[10]What is considered "domain independent" here is the means of constructing the policy.

[11]We can not exactly capture the TL-Plan knowledge in our policy language. Instead, we write policies that capture the knowledge but prune away some "bad" actions that TL-Plan might consider.

[12]For these stochastic domains we provide the heuristic (designed for deterministic domains) with a deterministic STRIPS domain approximation (using the mostly likely outcome of each action).

[13]The most complex blocks-world goal for RRL was to achieve **on**$(A, B)$ in an $n$ block environment. We consider blocks-world goals that involve all $n$ blocks.

## References

[1] Ricardo Aler, Daniel Borrajo, and Pedro Isasi. Using genetic programming to learn and improve control knowledge. *AIJ*, 141(1-2):29–56, 2002.

[2] Fahiem Bacchus. The AIPS '00 planning competition. *AI Magazine*, 22(3)(3):57–62, 2001.

[3] Fahiem Bacchus and Froduald Kabanza. Using temporal logics to express search control knowledge for planning. *AIJ*, 16:123–191, 2000.

[4] R. Bellman. *Dynamic Programming*. Princeton University Press, 1957.

[5] D. P. Bertsekas and J. N. Tsitsiklis. *Neuro-Dynamic Programming*. Athena Scientific, 1996.

[6] Craig Boutilier and Richard Dearden. Approximating value trees in structured dynamic programming. In Lorenza Saitta, editor, *ICML*, 1996.

[7] Craig Boutilier, Richard Dearden, and Moises Goldszmidt. Stochastic dynamic programming with factored representations. *AIJ*, 121(1-2):49–107, 2000.

[8] Craig Boutilier, Raymond Reiter, and Bob Price. Symbolic dynamic programming for first-order MDPs. In *IJCAI*, 2001.

[9] S. Dzeroski, L. DeRaedt & K. Driessens. Relational reinforcement learning. *MLJ*, 43:7–52, 2001.

[10] Tara A. Estlin and Raymond J. Mooney. Multi-strategy learning of search control for partial-order planning. In *AAAI*, 1996.

[11] Robert Givan, Thomas Dean, and Matt Greig. Equivalence notions and model minimization in Markov decision processes. *AIJ*, 147(1-2):163–223, 2003.

[12] Carlos Guestrin, Daphne Koller, and Ronald Parr. Max-norm projections for factored MDPs. In *IJCAI*, pages 673–680, 2001.

[13] Jorg Hoffmann and Bernhard Nebel. The FF planning system: Fast plan generation through heuristic search. *JAIR*, 14:263–302, 2001.

[14] R. Howard. *Dynamic Programming and Markov Decision Processes*. MIT Press, 1960.

[15] Yi-Cheng Huang, Bart Selman, and Henry Kautz. Learning declarative control rules for constraint-based planning. In *ICML*, pages 415–422, 2000.

[16] Michael J. Kearns, Yishay Mansour, and Andrew Y. Ng. A sparse sampling algorithm for near-optimal planning in large markov decision processes. *MLJ*, 49(2–3):193–208, 2002.

[17] Roni Khardon. Learning action strategies for planning domains. *AIJ*, 113(1-2):125–148, 1999.

[18] M. Lagoudakis and R. Parr. Reinforcement learning as classification: Leveraging modern classifiers. In *ICML*, 2003.

[19] Mario Martin and Hector Geffner. Learning generalized policies in planning domains using concept languages. In *KRR*, 2000.

[20] D. McAllester & R. Givan. Taxonomic syntax for 1st-order inference. *JACM*, 40:246–83, 1993.

[21] S. Minton. Quantitative results on the utility of explanation-based learning. In *AAAI*, 1988.

[22] S. Minton, editor. *Machine Learning Methods for Planning*. Morgan Kaufmann, 1993.

[23] S. Minton, J. Carbonell, C. A. Knoblock, D. R. Kuokka, O. Etzioni, and Y. Gil. Explanation-based learning: A problem solving perspective. *AIJ*, 40:63–118, 1989.

[24] G. Tesauro. Practical issues in temporal difference learning. *MLJ*, 8:257–277, 1992.

[25] G. Tesauro & G. Galperin. Online policy improvement via monte-carlo search. In *NIPS*, 1996.

[26] J. Tsitsiklis and B. Van Roy. Feature-based methods for large scale DP. *MLJ*, 22:59–94, 1996.

[27] M. Veloso, J. Carbonell, A. Perez, D. Borrajo, E. Fink, and J. Blythe. Integrating planning and learning: The PRODIGY architecture. *Journal of Experimental and Theoretical AI*, 7(1), 1995.

[28] S. Yoon, A. Fern, and R. Givan. Inductive policy selection for first-order MDPs. In *UAI*, 2002.

